# Hashing Hyperplane Queries to Near Points with Applications to Large-Scale Active Learning

**Prateek Jain**
Algorithms Research Group
Microsoft Research, Bangalore, India
`prajain@microsoft.com`

**Sudheendra Vijayanarasimhan**
Department of Computer Science
University of Texas at Austin
`svnaras@cs.utexas.edu`

**Kristen Grauman**
Department of Computer Science
University of Texas at Austin
`grauman@cs.utexas.edu`

## Abstract

We consider the problem of retrieving the database points nearest to a given *hyperplane* query without exhaustively scanning the database. We propose two hashing-based solutions. Our first approach maps the data to two-bit binary keys that are locality-sensitive for the angle between the hyperplane normal and a database point. Our second approach embeds the data into a vector space where the Euclidean norm reflects the desired distance between the original points and hyperplane query. Both use hashing to retrieve near points in sub-linear time. Our first method's preprocessing stage is more efficient, while the second has stronger accuracy guarantees. We apply both to pool-based active learning: taking the current hyperplane classifier as a query, our algorithm identifies those points (approximately) satisfying the well-known minimal distance-to-hyperplane selection criterion. We empirically demonstrate our methods' tradeoffs, and show that they make it practical to perform active selection with millions of unlabeled points.

## 1 Introduction

Efficient similarity search with large databases is central to many applications of interest, such as example-based learning algorithms, content-based image or audio retrieval, and quantization-based data compression. Often the search problem is considered in the domain of *point* data: given a database of vectors listing some attributes of the data objects, which points are nearest to a novel query vector? Existing algorithms provide efficient data structures for point-to-point retrieval tasks with various useful distance functions, producing either exact or approximate near neighbors while forgoing a brute force scan through all database items, e.g., [1, 2, 3, 4, 5, 6, 7].

By comparison, much less work considers how to efficiently handle instances more complex than points. In particular, little previous work addresses the *hyperplane-to-point* search problem: given a database of points, which are nearest to a novel hyperplane query? This problem is critical to pool-based active learning, where the goal is to request labels for those points that appear most informative. The widely used margin-based selection criterion of [8, 9, 10] seeks those points that are nearest to the current support vector machine's hyperplane decision boundary, and can substantially reduce total human annotation effort. However, for large-scale active learning, it is impractical to exhaustively apply the classifier to all unlabeled points at each round of learning; to exploit massive unlabeled pools, a fast (sub-linear time) hyperplane search method is needed.

To this end, we propose two solutions for approximate hyperplane-to-point search. For each, we introduce randomized hash functions that offer query times sub-linear in the size of the database, and provide bounds for the approximation error of the neighbors retrieved. Our first approach devises a two-bit hash function that is locality-sensitive for the angle between the hyperplane normal and a database point. Our second approach embeds the inputs such that the Euclidean distance reflects the hyperplane distance, thereby making them searchable with existing approximate nearest neighbor algorithms for vector data. While the preprocessing in our first method is more efficient, our second method has stronger accuracy guarantees.

We demonstrate our algorithms' significant practical impact for large-scale active learning with SVM classifiers. Our results show that our method helps scale-up active learning for realistic problems with massive unlabeled pools on the order of millions of examples.

## 2   Related Work

We briefly review related work on approximate similarity search, subspace search methods, and pool-based active learning.

**Approximate near-neighbor search.** For low-dimensional points, spatial decomposition and tree-based search algorithms can provide the exact neighbors in sub-linear time [1, 2]. While such methods break down for high-dimensional data, a number of *approximate* near neighbor methods have been proposed that work well with high-dimensional inputs. Locality-sensitive hashing (LSH) methods devise randomized hash functions that map similar points to the same hash buckets, so that only a subset of the database must be searched after hashing a novel query [3, 4, 5]. A related family of methods design Hamming space embeddings that can be indexed efficiently (e.g., [11, 12, 6]). However, in contrast to our approach, all such techniques are intended for vector/point data.

A few researchers have recently examined approximate search tasks involving subspaces. In [13], a Euclidean embedding is developed such that the norm in the embedding space directly reflects the principal angle-based distance between the original subspaces. After this mapping, one can apply existing approximate near-neighbor methods designed for points (e.g., LSH). We provide a related embedding to find the points nearest to the hyperplane; however, in contrast to [13], we provide LSH bounds, and our embedding is more compact due to our proposed sampling strategy. Another method to find the nearest subspace for a point query is given in [14], though it is limited to relatively low-dimensional data due to its preprocessing time/space requirement of $O(N^{d^2 \log N})$ and query time of $O(d^{10} \log N)$, where $N$ is the number of database points and $d$ is the dimensionality of the data. Further, unlike [13], that approach is restricted to point queries. Finally, a sub-linear time method to map a *line* query to its nearest points is derived in [15]. In contrast to all the above work, we propose specialized methods for the hyperplane search problem, and show that they handle high-dimensional data and large databases very efficiently.

**Margin-based active learning.** Existing active classifier learning methods for *pool-based* selection generally scan all database instances before selecting which to have labeled next.[1] One well-known and effective active selection criterion for support vector machines (SVMs) is to choose points that are nearest to the current separating hyperplane [8, 9, 10]. While simple, this criterion is intuitive, has theoretical basis in terms of rapidly reducing the version space [8], and thus is widely used in practice (e.g., [17, 18, 19]). Unfortunately, even for inexpensive selection functions, very large unlabeled datasets make the cost of exhaustively searching the pool impractical. Researchers have previously attempted to cope with this issue by clustering or randomly downsampling the pool [19, 20, 21, 22]; however, such strategies provide no guarantees as to the potential loss in active selection quality. In contrast, when applying our approach for this task, we can consider orders of magnitude fewer points when making the next active label request, yet guarantee selections within a known error of the traditional exhaustive pool-based technique.

**Other forms of approximate SVM training.** To avoid potential confusion, we note that our problem setting differs from both that considered in [23], where computational geometry insights are combined with the QP formulation for more efficient "core vector" SVM training, as well as that considered in [19], where a subset of *labeled* data points are selected for online LASVM training.

# 3 Approach

We consider the following retrieval problem. Given a database $\mathcal{D} = [\boldsymbol{x}_1, \ldots, \boldsymbol{x}_N]$ of $N$ points in $\mathbb{R}^d$, the goal is to retrieve the points from the database that are closest to a given *hyperplane* query whose normal is given by $\boldsymbol{w} \in \mathbb{R}^d$. We call this the *nearest neighbor to a query hyperplane* (NNQH) problem. Without loss of generality, we assume that the hyperplane passes through origin, and that each $\boldsymbol{x}_i$, $\boldsymbol{w}$ is unit norm. We see in later sections that these assumptions do not affect our solution.

The Euclidean distance of a point $\boldsymbol{x}$ to a given hyperplane $h_{\boldsymbol{w}}$ parameterized by normal $\boldsymbol{w}$ is:

$$d(h_{\boldsymbol{w}}, \boldsymbol{x}) = \|(\boldsymbol{x}^T \boldsymbol{w})\boldsymbol{w}\| = |\boldsymbol{x}^T \boldsymbol{w}|. \tag{1}$$

Thus, the goal for the NNQH problem is to identify those points $\boldsymbol{x}_i \in \mathcal{D}$ that minimize $|\boldsymbol{x}_i^T \boldsymbol{w}|$. Note that this is in contrast to traditional proximity problems, e.g., nearest or farthest neighbor retrieval, where the goal is to *maximize* $\boldsymbol{x}^T \boldsymbol{w}$ or $-\boldsymbol{x}^T \boldsymbol{w}$, respectively. Hence, existing approaches are not directly applicable to this problem.

We formulate two algorithms for NNQH. Our first approach maps the data to binary keys that are locality-sensitive for the angle between the hyperplane normal and a database point, thereby permitting sub-linear time retrieval with hashing. Our second approach computes a sparse Euclidean embedding for the query hyperplane that maps the desired search task to one handled well by existing approximate nearest-point methods.

In the following, we first provide necessary background on locality-sensitive hashing (LSH). The subsequent two sections describe each approach in turn, and Sec. 3.4 reviews their trade-offs. Finally, in Sec. 3.5, we explain how either method can be applied to large-scale active learning.

## 3.1 Background: Locality-Sensitive Hashing (LSH)

Informally, LSH [3] requires randomized hash functions guaranteeing that the probability of collision of two vectors is inversely proportional to their "distance", where "distance" is defined according to the task at hand. Since similar points are assured (w.h.p.) to fall into the same hash bucket, one need only search those database items with which a novel query collides in the hash table.

Formally, let $d(\cdot, \cdot)$ be a distance function over items from a set $S$, and for any item $p \in S$, let $B(p, r)$ denote the set of examples from $S$ within radius $r$ from $p$.

**Definition 3.1.** *[3] Let $h_{\mathcal{H}}$ denote a random choice of a hash function from the family $\mathcal{H}$. The family $\mathcal{H}$ is called $(r, r(1 + \epsilon), p_1, p_2)-$sensitive for $d(\cdot, \cdot)$ when, for any $q, p \in S$,*

- *if $p \in B(q, r)$ then $\Pr[h_{\mathcal{H}}(q) = h_{\mathcal{H}}(p)] \geq p_1$,*

- *if $p \notin B(q, r(1 + \epsilon))$ then $\Pr[h_{\mathcal{H}}(q) = h_{\mathcal{H}}(p)] \leq p_2$.*

For a family of functions to be useful, it must satisfy $p_1 > p_2$. A $k$-bit LSH function computes a hash "key" by concatenating the bits returned by a random sampling of $\mathcal{H}$: $g(p) = \left[ h_{\mathcal{H}}^{(1)}(p), h_{\mathcal{H}}^{(2)}(p), \ldots, h_{\mathcal{H}}^{(k)}(p) \right]$. Note that the probability of collision for close points is thus at least $p_1^k$, while for dissimilar points it is at most $p_2^k$. During a preprocessing stage, all database points are mapped to a series of $l$ hash tables indexed by independently constructed $g_1, \ldots, g_l$, where each $g_i$ is a $k$-bit function. Then, given a query $q$, an exhaustive search is carried out only on those examples in the union of the $l$ buckets to which $q$ hashes. These candidates contain the $(r, \epsilon)$-nearest neighbors (NN) for $q$, meaning if $q$ has a neighbor within radius $r$, then with high probability some example within radius $r(1 + \epsilon)$ is found.

In [3] an LSH scheme using projections onto single coordinates is shown to be locality-sensitive for the Hamming distance over vectors. For that hash function, $\rho = \frac{\log p_1}{\log p_2} \leq \frac{1}{1+\epsilon}$, and using $l = N^\rho$ hash tables, a $(1+\epsilon)$-approximate solution can be retrieved in time $O(N^{\frac{1}{(1+\epsilon)}})$. Related formulations and LSH functions for other distances have been explored (e.g., [5, 4, 24]). Our contribution is to define two locality-sensitive hash functions for the NNQH problem.

## 3.2 Hyperplane Hashing based on Angle Distance (H-Hash)

Recall that we want to retrieve the database vector(s) $\boldsymbol{x}$ for which $|\boldsymbol{w}^T\boldsymbol{x}|$ is minimized. If the vectors are unit norm, then this means that for the "good" (close) database vectors, $\boldsymbol{w}$ and $\boldsymbol{x}$ are almost perpendicular. Let $\theta_{\boldsymbol{x},\boldsymbol{w}}$ denote the angle between $\boldsymbol{x}$ and $\boldsymbol{w}$. We define the distance $d(\cdot,\cdot)$ in Definition 3.1 to reflect how far from perpendicular $\boldsymbol{w}$ and $\boldsymbol{x}$ are:

$$d_\theta(\boldsymbol{x},\boldsymbol{w}) = (\theta_{\boldsymbol{x},\boldsymbol{w}} - \pi/2)^2. \tag{2}$$

Consider the following two-bit function that maps two input vectors $\boldsymbol{a},\boldsymbol{b} \in \Re^d$ to $\{0,1\}^2$:

$$h_{\boldsymbol{u},\boldsymbol{v}}(\boldsymbol{a},\boldsymbol{b}) = [h_{\boldsymbol{u}}(\boldsymbol{a}), h_{\boldsymbol{v}}(\boldsymbol{b})] = [\text{sign}(\boldsymbol{u}^T\boldsymbol{a}), \text{sign}(\boldsymbol{v}^T\boldsymbol{b})], \tag{3}$$

where $h_{\boldsymbol{u}}(\boldsymbol{a}) = \text{sign}(\boldsymbol{u}^T\boldsymbol{a})$ returns 1 if $\boldsymbol{u}^T\boldsymbol{a} \geq 0$, and 0 otherwise, and $\boldsymbol{u}$ and $\boldsymbol{v}$ are sampled independently from a standard $d$-dimensional Gaussian, i.e., $\boldsymbol{u},\boldsymbol{v} \sim \mathcal{N}(0,I)$.

We define our **hyperplane hash** (H-Hash) function family $\mathcal{H}$ as:

$$h_\mathcal{H}(\boldsymbol{z}) = \begin{cases} h_{\boldsymbol{u},\boldsymbol{v}}(\boldsymbol{z},\boldsymbol{z}), & \text{if } \boldsymbol{z} \text{ is a database point vector,} \\ h_{\boldsymbol{u},\boldsymbol{v}}(\boldsymbol{z},-\boldsymbol{z}), & \text{if } \boldsymbol{z} \text{ is a query hyperplane vector.} \end{cases}$$

Next, we prove that this family of hash functions is locality-sensitive (Definition 3.1).

**Claim 3.2.** *The family* $\mathcal{H}$ *is* $\left(r, r(1+\epsilon), \frac{1}{4} - \frac{1}{\pi^2}r, \frac{1}{4} - \frac{1}{\pi^2}r(1+\epsilon)\right)$-*sensitive for the distance* $d_\theta(\cdot,\cdot)$, *where* $r,\epsilon > 0$.

*Proof.* Since the vectors $\boldsymbol{u}$, $\boldsymbol{v}$ used by hash function $h_{\boldsymbol{u},\boldsymbol{v}}$ are sampled independently, then for a query hyperplane vector $\boldsymbol{w}$ and a database point vector $\boldsymbol{x}$,

$$\begin{aligned} \Pr[h_\mathcal{H}(\boldsymbol{w}) = h_\mathcal{H}(\boldsymbol{x})] &= \Pr[h_{\boldsymbol{u}}(\boldsymbol{w}) = h_{\boldsymbol{u}}(\boldsymbol{x}) \text{ and } h_{\boldsymbol{v}}(-\boldsymbol{w}) = h_{\boldsymbol{v}}(\boldsymbol{x})], \\ &= \Pr[h_{\boldsymbol{u}}(\boldsymbol{w}) = h_{\boldsymbol{u}}(\boldsymbol{x})] \ \Pr[h_{\boldsymbol{v}}(-\boldsymbol{w}) = h_{\boldsymbol{v}}(\boldsymbol{x})]. \end{aligned} \tag{4}$$

Next, we use the following fact proven in [25],

$$\Pr[\text{sign}(\boldsymbol{u}^T\boldsymbol{a}) = \text{sign}(\boldsymbol{u}^T\boldsymbol{c})] = 1 - \frac{\theta_{\boldsymbol{a},\boldsymbol{c}}}{\pi}, \tag{5}$$

where $\boldsymbol{u}$ is sampled as defined above, and $\theta_{\boldsymbol{a},\boldsymbol{c}}$ denotes the angle between the two vectors $\boldsymbol{a}$ and $\boldsymbol{c}$.

Using (4) and (5), we get:

$$\Pr[h_\mathcal{H}(\boldsymbol{w}) = h_\mathcal{H}(\boldsymbol{x})] = \frac{\theta_{\boldsymbol{x},\boldsymbol{w}}}{\pi}\left(1 - \frac{\theta_{\boldsymbol{x},\boldsymbol{w}}}{\pi}\right) = \frac{1}{4} - \frac{1}{\pi^2}\left(\theta_{\boldsymbol{x},\boldsymbol{w}} - \frac{\pi}{2}\right)^2.$$

Hence, when $\left(\theta_{\boldsymbol{x},\boldsymbol{w}} - \frac{\pi}{2}\right)^2 \leq r$, $\Pr[h_\mathcal{H}(\boldsymbol{w}) = h_\mathcal{H}(\boldsymbol{x})] \geq \frac{1}{4} - \frac{r}{\pi^2} = p_1$. Similarly, for any $\epsilon > 0$ such that $\left(\theta_{\boldsymbol{x},\boldsymbol{w}} - \frac{\pi}{2}\right)^2 \geq r(1+\epsilon)$, $\Pr[h_\mathcal{H}(\boldsymbol{w}) = h_\mathcal{H}(\boldsymbol{x})] \leq \frac{1}{4} - \frac{r(1+\epsilon)}{\pi^2} = p_2$. $\square$

We note that unlike traditional LSH functions, ours are asymmetric. That is, to hash a database point $\boldsymbol{x}$ we use $h_{\boldsymbol{u},\boldsymbol{v}}(\boldsymbol{x},\boldsymbol{x})$, whereas to hash a query hyperplane $\boldsymbol{w}$, we use $h_{\boldsymbol{u},\boldsymbol{v}}(\boldsymbol{w},-\boldsymbol{w})$. The purpose of the two-bit hash is to constrain the angle with respect to both $\boldsymbol{w}$ and $-\boldsymbol{w}$, so that we do not simply retrieve examples for which we know only that $\boldsymbol{x}$ is $\pi/2$ *or less* away from $\boldsymbol{w}$.

With these functions in hand, we can now form hash keys by concatenating $k$ two-bit pairs from $k$ hash functions from $\mathcal{H}$, store the database points in the hash tables, and query with a novel hyperplane to retrieve its closest points (see Sec. 3.1).

The approximation guarantees and correctness of this scheme can be obtained by adapting the proof of Theorem 1 in [3] (see supplementary file). In particular, we can show that with high probability, our LSH scheme will return a point within a distance $(1+\epsilon)r$, where $r = \min_i d_\theta(\boldsymbol{x}_i,\boldsymbol{w})$, in time $O(N^\rho)$, where $\rho = \frac{\log p_1}{\log p_2}$. As $p_1 > p_2$, we have $\rho < 1$, i.e., the approach takes sub-linear time for all values of $r,\epsilon$. Furthermore, as $p_1 = \frac{1}{4} - \frac{r}{\pi^2}$, and $p_2 = \frac{1}{4} - \frac{r(1+\epsilon)}{\pi^2}$, $\rho$ can also be bounded as $\rho \leq \frac{1 - \log(1 - \frac{4r}{\pi^2})}{1 + \frac{\epsilon}{1 + \frac{\pi^2}{4r}}\log 4}$. Note that this bound for $\rho$ is dependent on $r$, and is more efficient for larger values of $r$. See the supplementary material for more discussion on the bound.

## 3.3 Embedded Hyperplane Hashing based on Euclidean Distance (EH-Hash)

Our second approach for the NNQH problem relies on a Euclidean embedding for the hyperplane and points. It offers stronger bounds than the above, but at the expense of more preprocessing.

Given a $d$-dimensional vector $\boldsymbol{a}$, we compute an embedding inspired by [13] that yields a $d^2$-dimensional vector by vectorizing the corresponding rank-1 matrix $\boldsymbol{a}\boldsymbol{a}^T$:

$$V(\boldsymbol{a}) = \text{vec}(\boldsymbol{a}\boldsymbol{a}^T) = \left[a_1^2, \, a_1 a_2, \dots, \, a_1 a_d, \, a_2^2, \, a_2 a_3, \dots, \, a_d^2\right], \tag{6}$$

where $a_i$ denotes the $i$-th element of $\boldsymbol{a}$. Assuming $\boldsymbol{a}$ and $\boldsymbol{b}$ to be unit vectors, the Euclidean distance between the embeddings $V(\boldsymbol{a})$ and $-V(\boldsymbol{b})$ is given by $\|V(\boldsymbol{a}) - (-V(\boldsymbol{b}))\|^2 = 2 + 2(\boldsymbol{a}^T\boldsymbol{b})^2$. Hence, minimizing the distance between the two embeddings is equivalent to minimizing $|\boldsymbol{a}^T\boldsymbol{b}|$, our intended function.

Given this, we define our **embedding-hyperplane hash** (EH-Hash) function family $\mathcal{E}$ as:

$$h_{\mathcal{E}}(\boldsymbol{z}) = \begin{cases} h_{\boldsymbol{u}}\left(V(\boldsymbol{z})\right), & \text{if } \boldsymbol{z} \text{ is a database point vector,} \\ h_{\boldsymbol{u}}\left(-V(\boldsymbol{z})\right), & \text{if } \boldsymbol{z} \text{ is a query hyperplane vector,} \end{cases}$$

where $h_{\boldsymbol{u}}(\boldsymbol{z}) = \text{sign}(\boldsymbol{u}^T \boldsymbol{z})$ is a one-bit hash function parameterized by $\boldsymbol{u} \sim \mathcal{N}(0, I)$.

**Claim 3.3.** *The family of functions $\mathcal{E}$ defined above is* $\left(r, r(1+\epsilon), \; \frac{1}{\pi}\cos^{-1}\sin^2(\sqrt{r}), \; \frac{1}{\pi}\cos^{-1}\sin^2(\sqrt{r(1+\epsilon)})\right)$*-sensitive for $d_\theta(\cdot, \cdot)$, where $r, \epsilon > 0$.*

*Proof.* Using the result of [25], for any vector $\boldsymbol{w}, \boldsymbol{x} \in \mathbb{R}^d$,

$$\Pr\left[\text{sign}\left(\boldsymbol{u}^T(-V(\boldsymbol{w}))\right) = \text{sign}\left(\boldsymbol{u}^T V(\boldsymbol{x})\right)\right] = 1 - \frac{1}{\pi}\cos^{-1}\left(\frac{-V(\boldsymbol{w})^T V(\boldsymbol{x})}{\|V(\boldsymbol{w})\| \; \|V(\boldsymbol{x})\|}\right), \tag{7}$$

where $\boldsymbol{u} \in \mathbb{R}^{d^2}$ is sampled from a standard $d^2$-variate Gaussian distribution, $\boldsymbol{u} \sim \mathcal{N}(0, I)$. Note that for any unit vectors $\boldsymbol{a}, \boldsymbol{b} \in \mathbb{R}^{d^2}$, $V(\boldsymbol{a})^T V(\boldsymbol{b}) = \text{Tr}(\boldsymbol{a}\boldsymbol{a}^T \boldsymbol{b}\boldsymbol{b}^T) = (\boldsymbol{a}^T\boldsymbol{b})^2 = \cos^2\theta_{\boldsymbol{a},\boldsymbol{b}}$.

Using (7) together with the definition of $h_{\mathcal{E}}$ above, given a hyperplane query $\boldsymbol{w}$ and database point $\boldsymbol{x}$ we have:

$$\Pr[h_{\mathcal{E}}(\boldsymbol{w}) = h_{\mathcal{E}}(\boldsymbol{x})] = 1 - \frac{1}{\pi}\cos^{-1}\left(-\cos^2(\theta_{\boldsymbol{x},\boldsymbol{w}})\right) = \cos^{-1}\left(\cos^2(\theta_{\boldsymbol{x},\boldsymbol{w}})\right)/\pi \tag{8}$$

Hence, when $(\theta_{\boldsymbol{x},\boldsymbol{w}} - \frac{\pi}{2})^2 \le r$,

$$\Pr[h_{\mathcal{E}}(\boldsymbol{w}) = h_{\mathcal{E}}(\boldsymbol{x})] \;\ge\; \frac{1}{\pi}\cos^{-1}\sin^2(\sqrt{r}) = p_1, \tag{9}$$

and $p_2$ is obtained similarly. $\qquad \square$

We observe that this $p_1$ behaves similarly to $2(\frac{1}{4} - \frac{r}{\pi^2})$. That is, as $r$ varies, EH-Hash's $p_1$ returns values close to twice those returned by H-Hash's $p_1$ (see plot illustrating this in supplementary file). Hence, the factor $\rho = \frac{\log p_1}{\log p_2}$ improves upon that of the previous section, remaining lower for lower values of $\epsilon$, and leading to better approximation guarantees. See supplementary material for a more detailed comparison of the two bounds.

On the other hand, EH-Hash's hash functions are significantly more expensive to compute. Specifically, it requires $O(d^2)$ time, whereas H-Hash requires only $O(d)$. To alleviate this problem, we use a form of randomized sampling when computing the hash bits for a query that reduces the time to $O(1/{\epsilon'}^2)$, for $\epsilon' > 0$. Our method relies on the following lemma, which states that sampling a vector $\boldsymbol{v}$ according to the weights of each element leads to good approximation to $\boldsymbol{v}^T\boldsymbol{y}$ for any vector $\boldsymbol{y}$ (with constant probability). Similar sampling schemes have been used for a variety of matrix approximation problems (see [26]).

**Lemma 3.4.** *Let $\boldsymbol{v} \in \mathbb{R}^d$ and define $p_i = v_i^2 / \|\boldsymbol{v}\|^2$. Construct $\tilde{\boldsymbol{v}} \in \mathbb{R}^d$ such that the $i$-th element is $v_i$ with probability $p_i$ and is $0$ otherwise. Select $t$ such elements using sampling with replacement. Then, for any $\boldsymbol{y} \in \mathbb{R}^d$, $\epsilon > 0$, $c \ge 1$, $t \ge \frac{c}{{\epsilon'}^2}$,*

$$\Pr[|\tilde{\boldsymbol{v}}^T\boldsymbol{y} - \boldsymbol{v}^T\boldsymbol{y}| \le \epsilon' \|\boldsymbol{v}\|^2 \|\boldsymbol{y}\|^2] > 1 - \frac{1}{c}. \tag{10}$$

We defer the proof to the supplementary material. The lemma implies that at query time our hash function $h_{\mathcal{E}}(\boldsymbol{w})$ can be computed while incurring a small additive error in time $O(\frac{1}{\epsilon'^2})$, by sampling its embedding $V(\boldsymbol{w})$ accordingly, and then cycling through only the non-zero indices of $V(\boldsymbol{w})$ to compute $\boldsymbol{u}^T(-V(\boldsymbol{w}))$. Note that we can substantially reduce the error in the hash function computation by sampling $O(\frac{1}{\epsilon'^2})$ elements of the vector $\boldsymbol{w}$ and then using $vec(\boldsymbol{w}\tilde{\boldsymbol{w}}^T)$ as the embedding for $\boldsymbol{w}$. However, in this case, the computational requirements increase to $O(\frac{d}{\epsilon'^2})$.

While one could alternatively use the Johnson-Lindenstrauss (JL) lemma to reduce the dimensionality of the embedding with random projections, doing so has two major difficulties: first, the $d-1$ dimensionality of a subspace represented by a hyperplane implies the random projection dimensionality must still be large for the JL-lemma to hold, and second, the projection dimension is dependent on the sum of the number of database points *and* query hyperplanes. The latter is problematic when fielding an arbitrary number of queries over time or storing a growing database of points—both properties that are intrinsic to our target active learning application. In contrast, our sampling method is instance-dependent and incurs very little overhead for computing the hash function.

**Comparison to [13].** Basri et al. define embeddings for finding nearest subspaces [13]. In particular, they define Euclidean embeddings for affine subspace queries and database points which could be used for NNQH, although they do not specifically apply it to hyperplane-to-point search in their work. Also, their embedding is not tied to LSH bounds in terms of the distance function (2), as we have shown above. Finally, our proposed instance-specific sampling strategy offers a more compact representation with the advantages discussed above.

### 3.4 Recap of the Hashing Approaches

To summarize, we presented two locality-sensitive hashing approaches for the NNQH problem. Our first H-Hash approach defines locality-sensitivity in the context of NNHQ, and then provides suitable two-bit hash functions together with a bound on retrieval time. Our second EH-Hash approach consists of a $d^2$-dimensional Euclidean embedding for vectors of dimension $d$ that in turn reduces NNHQ to the Euclidean space nearest neighbor problem, for which efficient search structures (including LSH) are available. While EH-Hash has better bounds than H-Hash, its hash functions are more expensive. To mitigate the expense for high-dimensional data, we use a well-justified heuristic where we randomly sample the given query embedding, reducing the query time to linear in $d$.

Note that both of our approaches attempt to minimize $d_\theta(\boldsymbol{w}, \boldsymbol{x})$ between the retrieved $\boldsymbol{x}$ and the hyperplane $\boldsymbol{w}$. Since that distance is only dependent on the *angle* between $\boldsymbol{x}$ and $\boldsymbol{w}$, any scaling of the vectors do not effect our methods, and we can safely treat the provided vectors to be unit norm.

### 3.5 Application to Large-Scale Active Learning

The search algorithms introduced above can be applied for any task fitting their query/database specifications. We are especially interested in their relevance for making active learning scalable.

A practical paradox with pool-based active learning algorithms is that their intended value—to reduce learning time by choosing informative examples to label first—conflicts with the real expense of applying them to very large "unprepared" unlabeled datasets. Generally methods today are tested in somewhat canned scenarios: the implementor has a moderately sized labeled dataset, and simply withholds the labels from the learner until a given point is selected, at which point the "oracle" reveals the label. In reality, one would like to deploy an active learner on a massive *truly* unlabeled data pool (e.g., all documents on the Web) and let it crawl for the instances that appear most valuable for the target classification task. The problem is that a scan of millions of points is rather expensive to compute exhaustively, and thus defeats the purpose of improving overall learning efficiency.

Our algorithms make it possible to benefit from *both* massive unlabeled collections as well as actively chosen label requests. We consider the "simple margin" selection criterion for linear SVM classifiers [8, 9, 10]. Given a hyperplane classifier and an unlabeled pool of vector data $\mathcal{U} = \{\boldsymbol{x}_1, \ldots, \boldsymbol{x}_N\}$, the point that minimizes the distance to the current decision boundary is selected for labeling: $\boldsymbol{x}^* = \operatorname{argmin}_{\boldsymbol{x}_i \in \mathcal{U}} |\boldsymbol{w}^T \boldsymbol{x}_i|$. Our two NNQH solutions supply exactly the hash functions needed to rapidly identify the next point to label: first we hash the unlabeled database into tables, and then at each active learning loop, we hash the current classifier $\boldsymbol{w}$ as a query.[2]

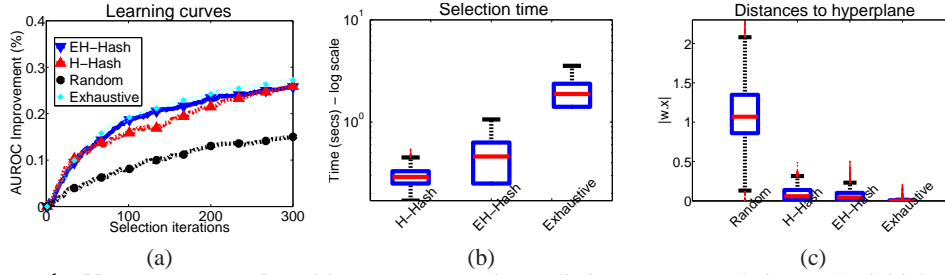

(a)                             (b)                             (c)

Figure 1: **Newsgroups results.** **(a)** Improvements in prediction accuracy relative to the initial classifier, averaged across all 20 categories and runs. **(b)** Time required to perform selection. **(c)** Value of $|\boldsymbol{w}^T\boldsymbol{x}|$ for the selected examples. Lower is better. Both of our approximate methods (H-Hash and EH-Hash) significantly outperform the passive baseline; they are nearly as accurate as ideal exhaustive active selection, yet require 1-2 orders of magnitude less time to select an example. (Best viewed in color.)

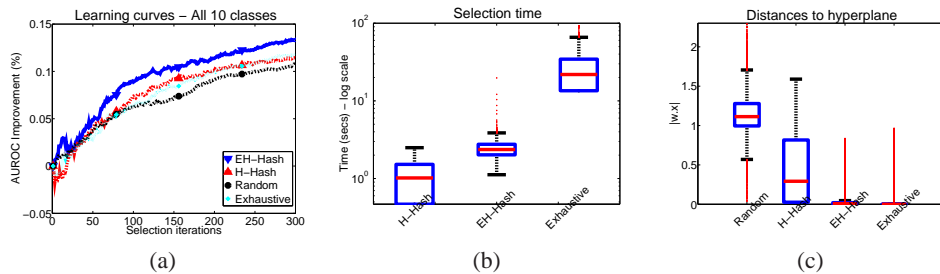

(a)                             (b)                             (c)

Figure 2: **CIFAR-10 results.** **(a)-(c)** Plotted as in above figure. Our methods compare very well with the significantly more expensive exhaustive baseline. Our EH-Hash provides more accurate selection than our H-Hash (see (c)), though requires noticeably more query time (see (b)).

## 4 Results

We demonstrate our approach applied to large-scale active learning tasks. We compare our methods (H-Hash in Sec. 3.2 and EH-Hash in Sec. 3.3) to two baselines: 1) passive learning, where the next label request is randomly selected, and 2) exhaustive active selection, where the margin criterion in (1) is computed over all unlabeled examples in order to find the true minimum. The main goal is to show our algorithms can retrieve examples nearly as well as the exhaustive approach, but with substantially greater efficiency.

**Datasets and implementation details.** We use three publicly available datasets. **20 Newsgroups** consists of 20,000 documents from 20 newsgroup categories. We use the provided 61,118-$d$ bag-of-words features, and a test set of 7,505. **CIFAR-10** [27] consists of 60,000 images from 10 categories. It is a manually labeled subset of the 80 Million Tiny Image dataset [28], which was formed by searching the Web for all English nouns and lacks ground truth labels. We use the provided train and test splits of $50K$ and $10K$ images, respectively. **Tiny-1M** consists of the first 1,000,000 (unlabeled) images from [28]. For both CIFAR-10 and Tiny-1M, we use the provided 384-$d$ GIST descriptors as features. For all datasets, we train a linear SVM in the one-vs-all setting using a randomly selected labeled set (5 examples per class), and then run active selection for 300 iterations. We average results across five such runs. We fix $k = 300$, $N^\rho = 500$, $\epsilon' = 0.01$.

**Newsgroups documents results.** Figure 1 shows the results on the 20 Newsgroups, starting with the learning curves for all four approaches (a). The active learners (exact and approximate) have the steepest curves, indicating that they are learning more effectively from the chosen labels compared to the random baseline. Both of our hashing methods perform similarly to the exhaustive selection, yet require scanning an order of magnitude fewer examples (b). Note, Random requires $\sim 0$ time. Fig. 1(c) shows the actual values of $|\boldsymbol{w}^T\boldsymbol{x}|$ for the selected examples over all iterations, categories, and runs; in line with our methods' guarantees, they select points close to those found with exhaustive search. We also observe the expected trade-off: H-Hash is more efficient, while EH-Hash provides better results (only slightly better for this smaller dataset).

**CIFAR-10 tiny image results.** Figure 2 shows the same set of results on the CIFAR-10. The trends are mostly similar to the above, although the learning task is more difficult on this data, narrowing the

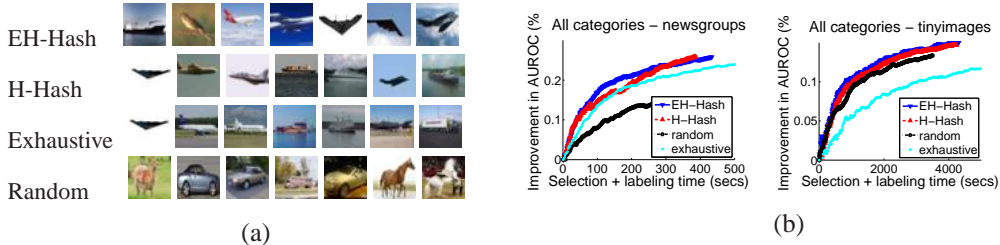

(a)

(b)

Figure 3: **(a)** First seven examples selected per method when learning the CIFAR-10 Airplane class. **(b)** Improvements in prediction accuracy as a function of the total time taken, including both selection and labeling time. By minimizing *both* selection and labeling time, our methods provide the best accuracy per unit time.

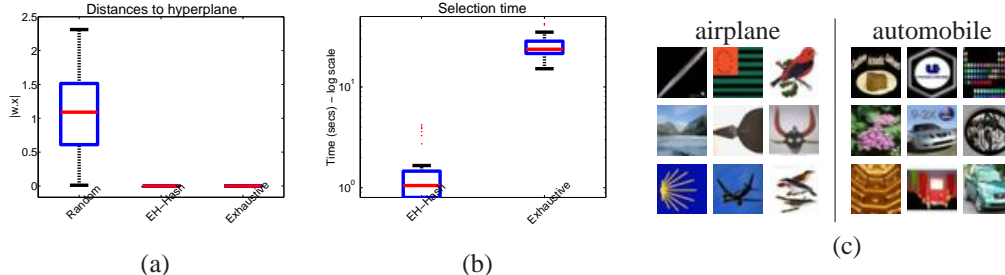

(a)                                    (b)                                                    (c)

Figure 4: **Tiny-1M results.** **(a)** Error of examples selected. **(b)** Time required. **(c)** Examples selected by EH-Hash among 1M candidates in the first nine iterations when learning the Airplane and Automobile classes.

margin between active and random. Averaged over all classes, we happen to outperform exhaustive selection (Fig. 2(a)); this can happen since there is no guarantee that the best active choice will help test accuracy, and it also reflects the wider variation across per-class results. The boxplots in (c) more directly show the hashing methods are behaving as expected. Both (b) and (c) illustrate their trade-offs: EH-Hash has stronger guarantees than H-Hash (and thus retrieves lower $w^T x$ values), but is more expensive. Figure 3(a) shows example image selection results; both exhaustive search and our hashing methods manage to choose images useful for learning about airplanes/non-airplanes.

Figure 3(b) shows the prediction accuracy plotted against the total time taken per iteration, which includes both *selection* and *labeling* time, for both datasets. We set the labeling time per instance to 1 and 5 seconds for the Newsgroups and Tiny image datasets, respectively. (Note, however, that these could vary in practice depending on the difficulty of the instance.) These results best show the advantage of our approximate methods: accounting for both types of cost inherent to training the classifier, they outperform both exhaustive and random selection in terms of the accuracy gains per unit time. While exhaustive active selection suffers because of its large *selection* time, random selection suffers because it wastes expensive *labeling* time on irrelevant examples. Our algorithms provide the best accuracy gains by minimizing both selection and labeling time.

**Tiny-1M results.** Finally, to demonstrate the practical capability of our hyperplane hashing approach, we perform active selection on the one million tiny image set. We initialize the classifier with 50 examples from CIFAR-10. The 1M set lacks any labels, making this a "live" test of active learning (we ourselves annotated whatever the methods selected). We use our EH-Hash method, since it offers stronger performance.

Even on this massive collection, our method's selections are very similar in quality to the exhaustive method (see Fig. 4(a)), yet require orders of magnitude less time (b). The images (c) show the selections made from this large pool during the "live" labeling test; among all one million unlabeled examples (nearly all of which likely belong to one of the other 1000s of *classes*) our method retrieves seemingly relevant instances. To our knowledge, this experiment exceeds any previous active selection results in the literature in terms of the scale of the unlabeled pool.

**Conclusions.** We introduced two methods for the NNQH search problem. Both permit efficient large-scale search for points near to a hyperplane, and experiments with three datasets clearly demonstrate the practical value for active learning with massive unlabeled pools. For future work, we plan to further explore more accurate hash-functions for our H-hash scheme and also investigate sublinear time methods for non-linear kernel based active learning.

This work is supported in part by DARPA CSSG, NSF EIA-0303609, and the Luce Foundation.

## Footnotes

[1]We consider only a specific hyperplane criterion in this paper; see [16] for an active learning survey.

[2]The SVM bias term is handled by appending points with a 1. Note, our approach assumes linear kernels.

# References

[1] J. Freidman, J. Bentley, and A. Finkel. An Algorithm for Finding Best Matches in Logarithmic Expected Time. *ACM Transactions on Mathematical Software*, 3(3):209–226, September 1977.

[2] J. Uhlmann. Satisfying General Proximity / Similarity Queries with Metric Trees. *Information Processing Letters*, 40:175–179, 1991.

[3] A. Gionis, P. Indyk, and R. Motwani. Similarity Search in High Dimensions via Hashing. In *Proceedings of the 25th Intl Conf. on Very Large Data Bases*, 1999.

[4] A. Andoni and P. Indyk. Near-Optimal Hashing Algorithms for Near Neighbor Problem in High Dimensions. In *FOCS*, 2006.

[5] M. Charikar. Similarity Estimation Techniques from Rounding Algorithms. In *STOC*, 2002.

[6] Y. Weiss, A. Torralba, and R. Fergus. Spectral Hashing. In *NIPS*, 2008.

[7] B. Kulis and K. Grauman. Kernelized Locality-Sensitive Hashing for Scalable Image Search. In *Proceedings of the IEEE International Conference on Computer Vision (ICCV)*, 2009.

[8] S. Tong and D. Koller. Support Vector Machine Active Learning with Applications to Text Classification. In *Proccedings of International Conference on Machine Learning*, 2000.

[9] G. Schohn and D. Cohn. Less is More: Active Learning with Support Vector Machines. In *Proccedings of International Conference on Machine Learning*, 2000.

[10] C. Campbell, N. Cristianini, and A. Smola. Query Learning with Large Margin Classifiers. In *Proccedings of International Conference on Machine Learning*, 2000.

[11] G. Shakhnarovich, P. Viola, and T. Darrell. Fast Pose Estimation with Parameter-Sensitive Hashing. In *Proceedings of the IEEE International Conference on Computer Vision (ICCV)*, 2003.

[12] R. Salakhutdinov and G. Hinton. Semantic Hashing. In *Proceedings of the SIGIR Workshop on Information Retrieval and Applications of Graphical Models*, 2007.

[13] R. Basri, T. Hassner, and L. Zelnik-Manor. Approximate Nearest Subspace Search. *PAMI*, 2010.

[14] A. Magen. Dimensionality Reductions that Preserve Volumes and Distance to Affine Spaces, and their Algorithmic Applications. In *Randomization and Approximation Techniques in Computer Science*, 2002.

[15] A. Andoni, P. Indyk, R. Krauthgamer, and H. L. Nguyen. Approximate Line Nearest Neighbor in High Dimensions. In *SODA*, 2009.

[16] B. Settles. Active Learning Literature Survey. TR 1648, University of Wisconsin, 2009.

[17] E. Chang, S. Tong, K. Goh, and C. Chang. Support Vector Machine Concept-Dependent Active Learning for Image Retrieval. In *IEEE Transactions on Multimedia*, 2005.

[18] M. K. Warmuth, J. Liao, G. Ratsch, M. Mathieson, S. Putta, and C. Lemmen. Active Learning with Support Vector Machines in the Drug Discovery Process. *J. Chem. Inf. Comput. Sci.*, 43:667–673, 2003.

[19] A. Bordes, S. Ertekin, J. Weston, and L. Bottou. Fast Kernel Classifiers with Online and Active Learning. *Journal of Machine Learning Research (JMLR)*, 6:1579–1619, September 2005.

[20] N. Panda, K. Goh, and E. Chang. Active Learning in Very Large Image Databases. *Journal of Multimedia Tools and Applications: Special Issue on Computer Vision Meets Databases*, 31(3), December 2006.

[21] W. Zhao, J. Long, E. Zhu, and Y. Liu. A Scalable Algorithm for Graph-Based Active Learning. In *Frontiers in Algorithmics*, 2008.

[22] R. Segal, T. Markowitz, and W. Arnold. Fast Uncertainty Sampling for Labeling Large E-mail Corpora. In *Conference on Email and Anti-Spam*, 2006.

[23] I. Tsang, J. Kwok, and P.-M. Cheung. Core Vector Machines: Fast SVM Training on Very Large Data Sets. *Journal of Machine Learning Research*, 6:363–392, 2005.

[24] P. Indyk and N. Thaper. Fast Image Retrieval via Embeddings. In *Intl Wkshp on Stat. and Comp. Theories of Vision*, 2003.

[25] M. Goemans and D. Williamson. Improved Approximation Algorithms for Maximum Cut and Satisfiability Problems Using Semidefinite Programming. *JACM*, 42(6):1115–1145, 1995.

[26] R. Kannan and S. Vempala. Spectral Algorithms. *Foundations and Trends in Theoretical Computer Science*, 4(3-4):157–288, 2009.

[27] A. Krizhevsky. Learning Multiple Layers of Features from Tiny Images. Technical report, University of Toronto, 2009.

[28] A. Torralba, R. Fergus, and W. T. Freeman. 80 million Tiny Images: a Large Dataset for Non-Parametric Object and Scene Recognition. *PAMI*, 30(11):1958–1970, 2008.

